# Sparsistent Learning of Varying-coefficient Models with Structural Changes

**Mladen Kolar, Le Song and Eric P. Xing** *
School of Computer Science, Carnegie Mellon University
{mkolar,lesong,epxing}@cs.cmu.edu

## Abstract

To estimate the changing structure of a varying-coefficient varying-structure (VCVS) model remains an important and open problem in dynamic system modelling, which includes learning trajectories of stock prices, or uncovering the topology of an evolving gene network. In this paper, we investigate sparsistent learning of a sub-family of this model — piecewise constant VCVS models. We analyze two main issues in this problem: inferring time points where structural changes occur and estimating model structure (i.e., model selection) on each of the constant segments. We propose a two-stage adaptive procedure, which first identifies jump points of structural changes and then identifies relevant covariates to a response on each of the segments. We provide an asymptotic analysis of the procedure, showing that with the increasing sample size, number of structural changes, and number of variables, the true model can be consistently selected. We demonstrate the performance of the method on synthetic data and apply it to the brain computer interface dataset. We also consider how this applies to structure estimation of time-varying probabilistic graphical models.

## 1 Introduction

Consider the following regression model:

$$Y_i = \mathbf{X}_i' \beta(t_i) + \epsilon_i, \quad i = 1, \ldots, n, \tag{1}$$

where the design variables $\mathbf{X}_i \in \mathbb{R}^p$ are *i.i.d.* zero mean random variables sampled at some conditions indexed by $i = 1, \ldots, n$, such as the prices of a set of stocks at time $i$, or the signals from some sensors deployed at location $i$; the noise $\epsilon_1, \ldots, \epsilon_n$ are *i.i.d.* Gaussian variables with variance $\sigma^2$ independent of the design variables; and $\beta(t_i) = (\beta_1(t_i), \ldots, \beta_p(t_i))' : [0, 1] \mapsto \mathbb{R}^p$ is a vector of unknown coefficient functions. Since the coefficient vector is a function of the conditions rather than a constant, such a model is called a *varying-coefficient model* [12]. Varying-coefficient models are a non-parametric extension to the linear regression models, which unlike other non-parametric models, assume that there is a linear relationship (generalizable to log-linear relationship) between the feature variables and the output variable, albeit a changing one. The model given in Eq. (1) has the flexibility of a non-parametric model and the interpretability of an ordinary linear regression.

Varying-coefficient models were popularized in the work of [9] and [16]. Since then, they have been applied to a variety of domains, including multidimensional regression, longitudinal and functional data analysis, and modeling problems in econometrics and finance, to model and predict time- or space- varying response to multidimensional inputs (see *e.g.* [12] for an overview.) One can easily imagine a more general form of such a model applicable to these domains, where both the coefficient value and the model structure change with values of other variables. We refer to this class of models as varying-coefficient varying-structure (VCVS) models. The more challenging problem of structure recovery (or model selection) under VCVS has started to catch attention very recently [1, 24].

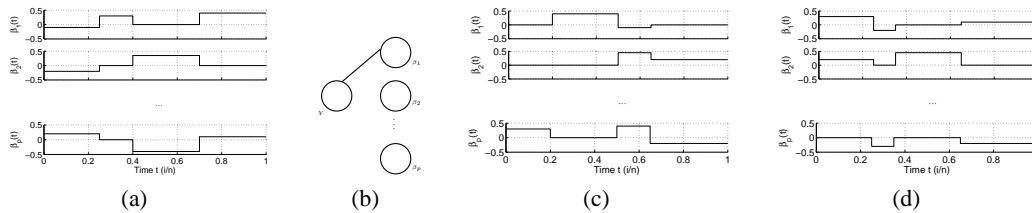

Figure 1: (a) Illustration of an VCVS as varying functions of time. The interval $[0, 1]$ is partitioned into $\{0, 0.25, 0.4, 0.7, 1\}$, which defines blocks on which the coefficient functions are constant. At different blocks only covariates with non-zero coefficient affect the response, *e.g.* on the interval $\mathcal{B}_2 = (0.25, 0.4)$ covariates $X_2$ and $X_p$ do not affect response. (b) Schematic representation of the covariates affecting the response during the second block in panel (a), which is reminiscent of neighborhood selection in graph structure learning. (c) and (d) Application of VCVS for graph structure estimation (see Section 7) of non-piecewise constant evolving graphs. Coefficients defining neighborhoods of different nodes can change on different partitions.

In this paper, we analyze VCVS as functions of time, and the main goal is to estimate the *dynamic structure* and *jump points* of the unknown vector function $\beta(t)$. To be more specific, we consider the case where the function $\beta(t)$ is time-varying, but piecewise constant (see Fig. 1), *i.e.*, there exists a partition $\mathcal{T} = \{T_1 = 0 < T_2 < \ldots < T_B = 1\}$, $1 < B \leq n$, of the time interval (scaled to) $[0, 1]$, such that $\beta(t) = \gamma_j$, $t \in [T_{j-1}, T_j)$ for some constant vectors $\gamma_j \in \mathbb{R}^p$, $j = 1, \ldots, B$. We refer to points $T_1, \ldots, T_B$ as jump points. Furthermore, we assume that at each time point $t_i$ only a few covariates affect the response, *i.e.*, the vector $\beta(t_i)$ is sparse. A good estimation procedure would be able to identify the correct partition of the interval $[0, 1]$ so that within each segment the coefficient function is constant. In addition, the procedure can identify active coefficients and their values within each segment, *i.e.*, the time-varying structure of the model. This estimation problem is particularly important in applications where one needs to uncover dynamic relational information or model structures from time series data. For example, one may want to infer at chosen time points the (changing) set of stocks that are predictive of a particular stock one has been holding from a time series of all stock prices; or to understand the evolving circuitry of gene regulation at different growth stages of an organism that determines the activity of a target gene based on other regulative genes, based on time series of microarray data. Another important problem is to identify structural changes in fields such as signal processing, EEG segmentation and analysis of seismic signals. In all these problems, the goal is not to estimate the optimum value of $\beta(t)$ for predicting $Y$, but to consistently uncover the zero and non-zero patterns in $\beta(t)$ at time points of interest that reveal the changing structure of the model. In this paper, we provide a new algorithm to achieve this goal, and a theoretical analysis that proves the asymptotic consistency of our algorithm.

Our problem is remotely related to, but very different from, earlier works on linear regression models with structural changes [4], and the problem of change-point detection (*e.g.* [19]), which can also be analyzed in the framework of varying-coefficient models. A number of existing methods are available to identify only one structural change in the data; in order to identify multiple changes these methods can be applied sequentially on smaller intervals that are assumed to harbor only one change [14]. Another common approach is to assume that there are $K$ changes and use Dynamic Programming to estimate them [4]. In this paper, we propose and analyze a penalized least squares approach, which automatically adapts to the unknown number of structural changes present in the data and performs the variable selection on each of the constant regions.

## 2 Preliminaries

For a varying-coefficient regression model described in Eq. (1) with structural changes, a reasonable estimator of the time-varying structure can be obtained by minimizing the so-called TESLA (temporally smoothed $L_1$-regularized regression) loss proposed in [1]: (for simplicity we suppress the sample-size notation $n$ in the regularization constants $\lambda^n = \{\lambda_1^n, \lambda_2^n\}$, but it should be clear that their values depend on $n$)

$$\hat{\beta}(t_1; \lambda), \ldots, \hat{\beta}(t_n; \lambda) = \arg\min_{\beta} \sum_{i=1}^{n} (Y_i - \mathbf{X}_i' \beta(t_i))^2 + 2\lambda_1 \sum_{i=1}^{n} ||\beta(t_i)||_1 + 2\lambda_2 \sum_{k=1}^{p} ||\beta_k||_{\text{TV}}, \quad (2)$$

where $||\cdot||_1$ denotes the $\ell_1$ norm, and $||\cdot||_{\text{TV}}$ denotes a total variation norm: $||\beta_k||_{\text{TV}} = \sum_{i=2}^{n} |\beta_k(t_i) - \beta_k(t_{i-1})|$. From the analysis of [20], it is known that each component function

$\beta_k$ can be chosen as a piecewise constant and right continuous function, *i.e.*, $\beta_k$ is a spline function, with potential jump points at observation times $t_i$, $i = 1, \ldots, n$. In this particular case, the total variation penalty defined above allows us to conceptualize $\beta_k$ as a vector in $\mathbb{R}^n$, whose components $\beta_{k,i} \equiv \beta_k(t_i)$ correspond to function values at $t_i, i = 1, \ldots, n$, but not as a function $[0, 1] \mapsto \mathbb{R}$. We continue to use the vector representation through the rest of the paper as it will simplify the notation.

The estimation problem defined in Eq. (2) has a few appealing properties. The objective function on the right-hand-side is convex and there exists a solution $\hat{\beta}$, which can be found efficiently using a standard convex optimization package. Furthermore, the penalty terms in Eq. (2) are constructed in a way to perform model selection. Observe that $\ell_1$ penalty encourages sparsity of the signal at each time point and enables a selection over the relevant coefficients; whereas the total variation penalty is used to partition the interval $[0, 1]$ so that $\hat{\beta}_k$ is constant within each segment. However, there are also some drawbacks of the procedure, as shown in Lemma 1 below.

Let's start with some notational clarifications. Let $\mathbf{X}$ denote the design matrix, input observation $\mathbf{X}_i$ at time $i$ corresponds to the $i$-th row in $\mathbf{X}$. For simplicity, we assume throughout the paper that $\mathbf{X}$ are normalized to have unit length columns, *i.e.*, each dimension has unit Euclidean norm. Let $\mathcal{B}_j$, $j = 1, \ldots, B$, denote the set of time points that fall into the interval $[T_{j-1}, T_j)$; when the meaning is clear from the context, we also use $\mathcal{B}_j$ as a shorthand of this interval. For example, $\mathbf{X}_{\mathcal{B}_j}$ and $Y_{\mathcal{B}_j}$ represent the submatrix of $\mathbf{X}$ and subvector of $Y$, respectively, that include elements only corresponding to time points within interval $\mathcal{B}_j$. For a given solution $\hat{\beta}$ to Eq. (2), there exists a block partition $\hat{\mathcal{T}} = \{\hat{T}_1, \ldots, \hat{T}_{\hat{B}}\}$ of $[0, 1]$ (possibly a trivial one) and unique vectors $\hat{\gamma}_j \in \mathbb{R}^p$, $j = 1, \ldots, \hat{B}$, such that $\hat{\beta}_{k,i} = \hat{\gamma}_{j,k}$ for $t_i \in \hat{\mathcal{B}}_j$. The set of relevant covariates during inverval $\mathcal{B}_j$, *i.e.*, the support of vector $\gamma_j$, is denoted as $S_{\mathcal{B}_j} = \{k \mid \gamma_{j,k} \neq 0\}$. Likewise we define $\hat{S}_{\hat{\mathcal{B}}_j}$ over $\hat{\gamma}_j$.

By construction, no consecutive vectors $\hat{\gamma}_j$ and $\hat{\gamma}_{j+1}$ are identical. Note that both the number of partitions $\hat{B} = |\hat{\mathcal{T}}|$, and the elements in the partition $\hat{\mathcal{T}}$, are random quantities. The following lemma characterizes the vectors $\hat{\gamma}_j$ using the subgradient equation of Eq. (2).

**Lemma 1** *Let $\hat{\gamma}_j$ and $\hat{\mathcal{B}}_j$, $j = 1, \ldots, \hat{B}$ be vectors and segments obtained from a minimizer of Eq. (2). Then each $\hat{\gamma}_j$ can be found as a solution to the subgradient equation:*

$$\mathbf{X}'_{\hat{\mathcal{B}}_j} \mathbf{X}_{\hat{\mathcal{B}}_j} \hat{\gamma}_j - \mathbf{X}'_{\hat{\mathcal{B}}_j} Y_{\hat{\mathcal{B}}_j} + \lambda_1 |\hat{\mathcal{B}}_j| \hat{s}_j^{(1)} + \lambda_2 \hat{s}_j^{(\mathrm{TV})} = 0, \tag{3}$$

*where*

$$\hat{s}_j^{(1)} \in \partial \, ||\hat{\gamma}_j||_1 = \mathrm{sign}(\gamma_j), \tag{4}$$

*by convention* $\mathrm{sign}(0) \in [-1, 1]$*, and $\hat{s}_j^{(\mathrm{TV})} \in \mathbb{R}^p$ such that*

$$\hat{s}_{1,k}^{(\mathrm{TV})} = \left\{ \begin{array}{ll} -1 & \textit{if } \hat{\gamma}_{2,k} - \hat{\gamma}_{1,k} > 0 \\ 1 & \textit{if } \hat{\gamma}_{2,k} - \hat{\gamma}_{1,k} < 0 \end{array} \right. \quad , \quad \hat{s}_{\hat{B},k}^{(\mathrm{TV})} = \left\{ \begin{array}{ll} 1 & \textit{if } \hat{\gamma}_{\hat{B},k} - \hat{\gamma}_{\hat{B}-1,k} > 0 \\ -1 & \textit{if } \hat{\gamma}_{\hat{B},k} - \hat{\gamma}_{\hat{B}-1,k} < 0 \end{array} \right. \tag{5}$$

*and, for $1 < j < \hat{B}$,*

$$\hat{s}_{j,k}^{(\mathrm{TV})} = \left\{ \begin{array}{ll} 2 & \textit{if } \hat{\gamma}_{j+1,k} - \hat{\gamma}_{j,k} > 0, \hat{\gamma}_{j,k} - \hat{\gamma}_{j-1,k} < 0 \\ -2 & \textit{if } \hat{\gamma}_{j+1,k} - \hat{\gamma}_{j,k} < 0, \hat{\gamma}_{j,k} - \hat{\gamma}_{j-1,k} > 0 \\ 0 & \textit{if } (\hat{\gamma}_{j,k} - \hat{\gamma}_{j-1,k})(\hat{\gamma}_{j+1,k} - \hat{\gamma}_{j,k}) = 1. \end{array} \right. \tag{6}$$

Lemma 1 does not provide a practical way to estimate $\hat{\beta}^{\mathrm{TV}}$, but it does characterize a solution. From Eq. (3) we can see that the coefficients in each of the estimated blocks are biased by two terms coming from the $\ell_1$ and $||\cdot||_{\mathrm{TV}}$ penalties. The larger the estimated segments, the smaller the relative influence of the bias from the total variation, while the magnitude of the bias introduced by the $\ell_1$ penalty is uniform across different segments. The additional bias coming from the total variation penalty was also noted in the problem of signal denoising [23]. In the next section, we introduce a two step procedure which alleviate this effect.

## 3 A two-step procedure for estimating time-varying structures

In this section, we propose a new algorithm for estimating the time-varying structure of the varying-coefficient model in Eq. (1), which does not suffer from the bias introduced by minimizing the objective in Eq. (2). The algorithm is a two-step procedure summarized as follows:

1. Estimate the block partition $\hat{\mathcal{T}}$, on which the coefficient vector is constant within each block. This can be obtained by minimizing the following objective:

$$\sum_{i=1}^{n} \left(Y_i - \mathbf{X}_i' \beta(t_i)\right)^2 + 2\lambda_2 \sum_{k=1}^{p} ||\beta_k||_{\text{TV}}, \tag{7}$$

which we refer to as a *temporal difference* (TD) regression for reasons that will be clear shortly. We will employ a TD-transformation to Eq. (7) and turn it into an $\ell_1$-regularized regression problem, and solve it using the randomized Lasso. Details of the algorithm and how to extract $\hat{\mathcal{T}}$ from the TD-estimate will be given shortly.

2. For each block of the partition, $\hat{\mathcal{B}}_j, 1 \le j \le \hat{B}$, estimate $\hat{\gamma}_j$ by minimizing the Lasso objective within the block:

$$\hat{\gamma}_j = \operatorname*{argmin}_{\gamma \in \mathbb{R}^p} \sum_{t_i \in \hat{\mathcal{B}}_j} (Y_i - \mathbf{X}_i' \gamma)^2 + 2\lambda_1 ||\gamma||_1. \tag{8}$$

We name this procedure TDB-Lasso (or TDBL), after the two steps (TD randomized Lasso, and Lasso within Blocks) given above. The advantage of the TDB-Lasso compared to a minimizer of Eq. (2) comes from decoupling the interactions between the $\ell_1$ and TV penalties (note that the two procedures result in different estimates). Now we discuss step 1 in detail; step 2 is straightforward using a standard Lasso toolbox.

To obtain a consistent estimate of $\hat{\mathcal{T}}$ from the TD-regression in Eq. (7), we can transform Eq. (7) into an equivalent $\ell_1$ penalized regression problem, which allows us to cast the $\hat{\mathcal{T}}$ estimation problem as a feature selection problem. Let $\beta_{k,i}^{\dagger}$ denote the *temporal difference* between the regression coefficients corresponding to the same covariate $k$ at successive time points $t_{i-1}$ and $t_i$: $\beta_{k,i}^{\dagger} \equiv \beta_k(t_i) - \beta_k(t_{i-1}), k = 1, \ldots, p, i = 1, \ldots, n$ with $\beta_k(t_0) = 0$, by convention. It can be shown the model in Eq. (1) can be expressed as $Y^{\dagger} = \mathbf{X}^{\dagger} \beta^{\dagger} + \epsilon^{\dagger}$, where $Y^{\dagger} \in \mathbb{R}^n$ is a transformed vector of the TDs of responses, *i.e.*, each element $Y_i^{\dagger} \equiv Y_i - Y_{i-1}$; $\mathbf{X}^{\dagger} = (\mathbf{X}_1^{\dagger}, \ldots, \mathbf{X}_p^{\dagger}) \in \mathbb{R}^{n \times np}$ is the transformed design matrix with lower triangular matrices $\mathbf{X}_k^{\dagger} \in \mathbb{R}^{n \times n}$ corresponding to TD features computed from the covariates; $\epsilon^{\dagger} \in \mathbb{R}^n$ is the transformed TD-error vector; and $\beta^{\dagger} \in \mathbb{R}^{np}$ is a vector obtained by stacking TD-coefficient vectors $\beta_k^{\dagger}$. (See Appendix for more details of the transformation.) Note that the elements of the vector $\epsilon^{\dagger}$ are not *i.i.d.* any more. Using the transformation above, the estimation problem defined on objective Eq. (7) can be expressed in the following matrix form:

$$\hat{\beta}^{\dagger} = \operatorname*{argmin}_{\beta \in \mathbb{R}^{np}} ||Y^{\dagger} - \mathbf{X}^{\dagger} \beta^{\dagger}||_2^2 + 2\lambda_2 ||\beta^{\dagger}||_1. \tag{9}$$

This transformation was proposed in [8] in the context of one-dimensional signal denoising, however, we are interested in the estimation of jump points in the context of time-varying coefficient model.

The estimator defined in Eq. (9) is not robust with respect to small perturbations of data, *i.e.*, small changes of variables $\mathbf{X}_i$ or $Y_i$ would result in a different $\hat{\mathcal{T}}$. To deal with the problem of robustness, we employed the *stability selection* procedure of [22] (see also the bootstrap Lasso [2], however, we have decided to use the stability selection because of the weaker assumptions). The stability selection approach to estimating the jump-points is comprised of two main components: i) simulating multiple datasets using bootstrap, and ii) using the randomized Lasso outlined in Algorithm 1 (see also Appendix) to solve (9). While the bootstrap step improves the robustness of the estimator, the randomized Lasso weakens the conditions under which the estimator $\hat{\beta}^{\dagger}$ selects exactly the true features.

Let $\{\hat{\beta}_b^{\dagger}, \hat{\mathcal{J}}_b^{\dagger}\}_{b=1}^{M}$ represent the set of estimates and their supports (*i.e.*, index of non-zero elements) obtained by minimizing (9) for each of the $M$ bootstrapped datasets. We obtain a stable estimate of the support by selecting variables that appear in multiple supports

$$\hat{\mathcal{J}}^{\tau} = \{k \mid \frac{\sum_{b=1}^{M} \mathbb{1}\{k \in \hat{\mathcal{J}}_b^{\dagger}\}}{M} \ge \tau\}, \tag{10}$$

which is then used to obtain the block partition estimate $\hat{\mathcal{T}}$. The parameter $\tau$ is a tuning parameter that controls the number of falsely identified jump points.

---
**Algorithm 1** Randomized Lasso
---
**Input**: Dataset $\{\mathbf{X}_i, Y_i\}_{i=1}^n$ $\mathbf{X}_i \in \mathbb{R}^p$, penalty parameter $\lambda$, weakness parameter $\alpha \in (0, 1]$
**Output**: Estimate $\hat{\beta} \in \mathbb{R}^p$, support $\hat{S}$
  1: Choose randomly $p$ weights $\{W_k\}_{k=1}^p$ from interval $[\alpha, 1]$
  2: $\hat{\beta} = \text{argmin}_{\beta \in \mathbb{R}^p} \sum_{i=1}^n (Y_i - \mathbf{X}_i \beta)^2 + 2\lambda \sum_{k=1}^p \frac{|\beta_k|}{W_k}$
  3: $\hat{S} = \{k \mid \hat{\beta}_k \neq 0\}$
---

## 4 Theoretical analysis

We provide a theoretical analysis of TDB-Lasso, and show that under certain conditions both the jump points and structure of VCVS can be consistently estimated. Proofs are deferred to Appendix.

### 4.1 Estimating jump points

We first address the issue of estimating jump points by analyzing the transformed TD-regression problem Eq. (9) and its feature selection properties. The feature selection using $\ell_1$ penalization has been analyzed intensively over the past few years and we can adapt some of the existing results to the problem at hand. To prove that all the jump points are included in $\hat{\mathcal{J}}^\tau$, we first state a *sparse eigenvalue condition* on the design (*e.g.* [6]). The minimal and maximal sparse eigenvalue, for matrix $\mathbf{X} \in \mathbb{R}^{n \times p}$, are defined as

$$\varphi_{\min}(k, \mathbf{X}) := \inf_{a \in \mathbb{R}^p, ||a||_0 \leq k} \frac{||\mathbf{X}a||_2}{||a||_2}, \qquad \varphi_{\max}(k, \mathbf{X}) := \sup_{a \in \mathbb{R}^p, ||a||_0 \leq k} \frac{||\mathbf{X}a||_2}{||a||_2}, \quad k \leq p. \quad (11)$$

Note that in Eq. (11) eigenvalues are computed over submatrices of size $k$ (i.e., due to the constraint on $a$ by the $||\cdot||_0$ norm). We can now express the sparse eigenvalues condition on the design.

**A1:** Let $\mathcal{J}^\dagger$ be the true support of $\beta^\dagger$ and $J = |\mathcal{J}^\dagger|$. There exist some $C > 1$ and $\kappa \geq 10$ such that

$$\frac{\varphi_{\max}(CJ^2, \mathbf{X}^\dagger)}{\varphi_{\min}^{3/2}(CJ^2, \mathbf{X}^\dagger)} < \sqrt{C}/\kappa. \quad (12)$$

This condition guarantees a correlation structure between TD-transformed covariates that allows for detection of the jump points. Comparing to the *irrepresentible condition* [30, 21, 27], necessary for the ordinary Lasso to perform feature selection, condition A1 is much weaker [22] and is sufficient for the randomized Lasso to select the relevant feature with high probability (see also [26]).

**Theorem 1** *Let A1 be satisfied; and let the weakness $\alpha$ be given as $\alpha^2 = \nu\varphi_{\min}(CJ^2, \mathbf{X}^\dagger)/(CJ^2)$, for any $\nu \in (7/\kappa, 1/\sqrt{2})$. If the minimum size of the jump is bounded away from zero as*

$$\min_{k \in \mathcal{J}^\dagger} |\beta_k^\dagger| \geq 0.3(CJ)^{3/2}\lambda_{\min}, \quad (13)$$

*where $\lambda_{\min} = 2\sigma^\dagger(\sqrt{C}J + 1)\sqrt{\frac{\log np}{n}}$ and $\sigma^{\dagger 2} \geq Var(Y_i^\dagger)$, for $np > 10$ and $J \geq 7$, there exists some $\delta = \delta_J \in (0, 1)$ such that for all $\tau \geq 1 - \delta$, the collection of the estimated jump points $\hat{\mathcal{J}}^\tau$ satisfies,*

$$\mathbb{P}(\hat{\mathcal{J}}^\tau = \mathcal{J}^\dagger) \geq 1 - 5/np. \quad (14)$$

**Remark:** Note that Theorem 1 gives conditions under which we can recover every jump point in every covariates. In particular, there are no assumptions on the number of covariates that change values at a jump point. Assuming that multiple covariates change their values at a jump point, we could further relax the condition on the minimal size of a jump given in Eq. (13). It was also pointed to us that the framework of [18] may be a more natural way to estimate jump points.

### 4.2 Identifying correct covariates

Now we address the issue of selecting the relevant features for every estimated segment. Under the conditions of Theorem 1, correct jump points will be detected with probability arbitrarily close to 1. That means under the assumption A1, we can run the regular Lasso on each of the estimated segments to select the relevant features therein. We will assume that the mutual coherence condition [10] holds for each segment $\mathcal{B}_j$. Let $\boldsymbol{\Sigma}^j = \frac{1}{|\mathcal{B}_j|} \sum_{i \in \mathcal{B}_j} \mathbf{X}_i' \mathbf{X}_i$, with $\sigma_{kl}^j = (\boldsymbol{\Sigma}^j)_{k,l}$.

**A2:** We assume there is a constant $0 < d \leq 1$ such that

$$\mathbb{P}\left(\max_{k \in S_{\mathcal{B}_j}, l \neq k} \left\{|\sigma_{kl}^j| \leq \frac{d}{|S_{\mathcal{B}_j}|}\right\}\right) = 1. \quad (15)$$

The assumption A2 is a mild version of the mutual coherence condition used in [7], which is necessary for identification of the relevant covariates in each segment. Let $\hat{\gamma}_j$, $k = 1, \dots, \hat{B}_n$ denote the Lasso estimates for each segment obtained by minimizing (8).

**Theorem 2** *Let A2 be satisfied. Also, assume that the conditions of Theorem 1 are satisfied. Let $K = \max_{1 \leq j \leq B} ||\gamma_j||_0$ be the upper bound on the number of features in segments and let $L$ be an upper bound on elements of* $\mathbf{X}$*. Let $\rho = \min_{1 \leq j \leq B} |\mathcal{B}_j|$ denote the number of samples in the smallest segment. Then for a sequence $\delta = \delta_n \rightarrow 0$,*

$$\lambda_1 \geq 4L\sigma\sqrt{\frac{\ln \frac{2Kp}{\delta}}{\rho}} \vee 8L\frac{\ln \frac{4Kp}{\delta}}{\rho} \quad and \quad \min_{1 \leq j \leq B} \min_{k \in S_{\mathcal{B}_j}} |\gamma_{j,k}| \geq 2\lambda_1,$$

*we have*

$$\lim_{n \rightarrow \infty} \mathbb{P}(\hat{B} = B) = 1, \quad (16)$$

$$\lim_{n \rightarrow \infty} \max_{1 \leq j \leq B} \mathbb{P}(||\hat{\gamma}_j - \gamma_j||_1 = 0) = 1, \quad (17)$$

$$\lim_{n \rightarrow \infty} \min_{1 \leq j \leq B} \mathbb{P}(\hat{S}_{\mathcal{B}_j} = S_{\mathcal{B}_j}) = 1. \quad (18)$$

Theorem 2 states that asymptotically, the two stage procedure estimates the correct model, *i.e.*, it selects the correct jump points and for each segment between two jump points it is able to select the correct covariates. Furthermore, we can conclude that the procedure is consistent.

## 5 Practical considerations

As in standard Lasso, the regularization parameters in TDB-Lasso need to be tuned appropriately to attain correct structural recovery. The TD regression procedure requires three parameters: the penalty parameter $\lambda_2$, cut-off parameter $\tau$, and weakness parameter $\alpha$. From our empirical experience, the recovered set of jump points $\hat{\mathcal{T}}$ vary very little with respect to these parameters in a wide range. The result of Theorem 1 is valid as long as $\lambda_2$ is larger than $\lambda_{\min}$ given in the statement of the theorem. Theorem 1 in [22] gives a way to select the cutoff $\tau$ while controlling the number of falsely included jump points. Note that this relieves users from carefully choosing the range of parameter $\lambda_2$, which is challenging. The weakness parameter can be chosen in quite a large interval (see Appendix on the randomized Lasso) and we report our results using the values $\alpha = 0.6$.

In the second step of the algorithm, the ordinary Lasso minimizes Eq. (8) on each estimated segment to select relevant variables, which requires a choice of the penalty parameter $\lambda_1$. We do so by minimizing the BIC criterion [25].

In practice, one cannot verify assumptions A1 and A2 on real datasets. In cases where the assumptions are violated, the resulting set of estimated jump points is larger than the true set $\mathcal{T}$, *e.g.* the points close to the true jump points get included into the resulting estimate $\hat{\mathcal{T}}$. We propose to use an *ad hoc* heuristic to refine the initially selected set of jump points. A commonly used procedure for estimation of linear regression models with structural changes [3] is a dynamic programming method that considers a possible structural change at every location $t_i$, $i = 1, \dots, n$, with a computational complexity of $\mathcal{O}(n^2)$ (see also [15]). We modify this method to consider jump points only in the estimated set $\hat{\mathcal{T}}$ and thus considerably reducing the computational complexity to $\mathcal{O}(|\hat{\mathcal{T}}|^2)$, since $|\hat{\mathcal{T}}| \ll n$. The algorithm effectively chooses a subset $\tilde{\mathcal{T}} \subseteq \hat{\mathcal{T}}$ of size $\hat{B}$ that minimizes the BIC objective.

## 6 Experiments on Synthetic Data

We compared the TDB-Lasso on synthetic data with commonly used methods for estimating VCVS models. The synthetic data was generated as follows. We varied the sample size from $n = 100$

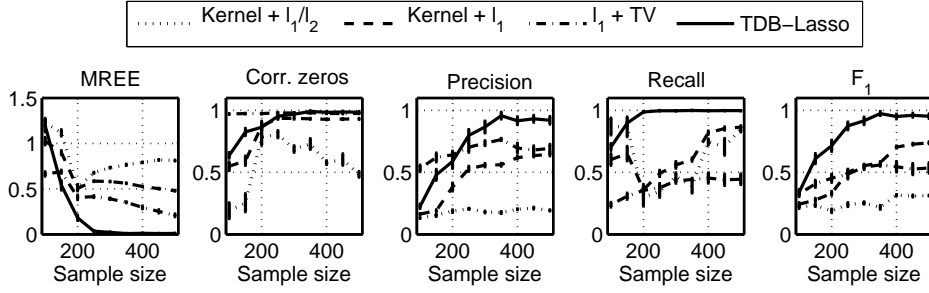

Figure 2: Comparison results of different estimation procedures on a synthetic dataset.

to $500$ time points, and fixed the number of covariates is fixed to $p = 20$. The block partition was generated randomly and consists of ten blocks with minimum length set to 10 time points. In each of the block, only 5 covariates out of 20 affected the response. Their values were uniformly at random drawn from $[-1, -0.1] \cup [0.1, 1]$. With this configuration, a dataset was created by randomly drawing $\mathbf{X}_i \sim N(0, I_p)$, $\epsilon_i \sim N(0, 1.5^2)$ and computing $Y_i = \mathbf{X}_i \beta(t_i) + \epsilon_i$ for $i = 1, \dots, n$. For each sample size, we independently generated 100 datasets and report results averaged over them.

A simple local regression method [13], which is commonly used for estimation in varying coefficient models, was used as the simplest baseline for comparing the relative performance of estimation. Our first competitor is an extension of the baseline, which uses the following estimator [28]:

$$\min_{\beta \in \mathbb{R}^{p \times n}} \sum_{i'=1}^{n} \sum_{i=1}^{n} (Y_i - \mathbf{X}'_i \beta_{i'})^2 K_h(t_{i'} - t_i) + \sum_{j=1}^{p} \lambda_j \sqrt{\sum_{i'=1}^{n} \beta_{i',j}^2}, \qquad (19)$$

where $K_h(\cdot) = \frac{1}{h} K(\cdot/h)$ is the kernel function. We will call this method "Kernel $\ell_1/\ell_2$". Another competitor uses the $\ell_1$ penalized local regression independently at each time point, which leads to the following estimator of $\beta(t)$,

$$\min_{\beta \in \mathbb{R}^p} \sum_{i=1}^{n} (Y_i - \mathbf{X}'_i \beta)^2 K_h(t_i - t) + \sum_{j=1}^{p} \lambda_j |\beta_j|. \qquad (20)$$

We call this method "Kernel $\ell_1$". The difference between the two methods is that "Kernel $\ell_1/\ell_2$" biases certain covariates toward zero at every time point, based on global information; whereas "Kernel $\ell_1$" biases covariates toward zero only based on local information. The final competitor is chosen to be the minimizer of Eq. (2) [1], which we call "$\ell_1$ + TV". The bandwidth parameter for "Kernel $\ell_1$" and "Kernel $\ell_1/\ell_2$" is chosen using a generalized cross validation of a non-penalized estimator. The penalty parameters $\lambda_j$ are chosen according to the BIC criterion [28]. For the "$\ell_1$ + TV" method, we optimize the BIC criterion over a two-dimensional grid of values for $\lambda_1$ and $\lambda_2$.

We report the relative estimation error, $\text{REE} = 100 \times \frac{\sum_{i=1}^{n} \sum_{j=1}^{p} |\hat{\beta}_{i,j} - \beta^*_{i,j}|}{\sum_{i=1}^{n} \sum_{j=1}^{p} |\tilde{\beta}_{i,j} - \beta^*_{i,j}|}$, where $\tilde{\beta}$ is the baseline local linear estimator, as a measure of estimation accuracy. To asses the performance of the model selection, we report precision, recall and their harmonic mean $F_1$ measure when estimating the relevant covariates at each time point and the percentage of correctly identified irrelevant covariates.

From the experimental results, summarized in Fig. 2, we can see that the TDB-Lasso succeeds in recovering the true model as the sample size increases. It also estimates the coefficient values with better accuracy than the other methods. It worth noting that the "Kernel + $\ell_1$" performs better than the "Kernel + $\ell_1/\ell_2$" approach, which is due to the violation of the assumptions made in [28]. The "$\ell_1$ + TV" performs better than the local linear regression approaches, however, the method gets very slow for the larger values of the sample size and it requires selecting two tuning parameters, which makes it quite difficult to use. We conjecture that the "$\ell_1$ + TV" and TDB-Lasso have similar asymptotic properties with respect to model selection, however, from our numerical experiments we can see that for finite sample data, the TDB-Lasso performs better.

## 7 Application to Time-varying Graph Structure Estimation

An interesting application of the TDB-Lasso is in structural estimation of time-varying undirected graphical models [1, 17]. A graph structure estimation can be posed as a neighborhood selection

problem, in which neighbors of each node are estimated independently. Neighborhood selection in the time-varying Gaussian graphical models (GGM) is equivalent to model selection in VCVS, where value of one node is regressed to the rest of nodes. The regression problem for each node can be solved using the TDB-Lasso. Graphs estimated in this way will have neighborhoods of each node that are constant on a partition, but the graph as a whole changes more flexibly (Fig. 1b-d).

The graph structure estimation using the TDB-Lasso is demonstrated on a real dataset of electroencephalogram (EEG) measurements. We use the brain computer interface (BCI) dataset IVa from [11] in which the EEG data is collected from 5 subjects, who were given visual cues based on which they were required to imagine right hand or right foot for 3.5s. The measurement was performed when the visual cues were presented on the

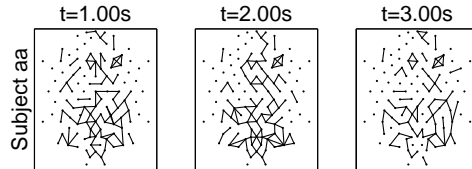

Figure 3: Brain interactions for the subject 'aa' when presented with visual cues of the class 1

screen (280 times), intermitted by periods of random length in which the subject could relax. We use the down-sampled data at 100Hz. Fig. 3 gives a visualization of the brain interactions over the time of the experiment for the subject 'aa' while presented with visual cues for the class 1 (right hand). Estimated graphs of interactions between different parts of the brain for other subjects and classes are given in Appendix due to the space limit.

We also want to study whether the estimated time-varying network are discriminative features for classifying the type of imaginations in the EEG signal. For this purpose, we perform unsupervised clustering of EEG signals using the time-varying networks and study whether the grouping correspond to the true grouping according to imagination label. We estimate a time-varying GGM using the TDB-Lasso for each visual cue and cluster the graphs using the spectral K-means clustering [29] (using a linear kernel on the coefficients to measure similarity). Each cluster is labeled according to the majority of points it contains. Finally, each cue if classified based on labels of the time-points that it contains. Table 1 summarizes the classification accuracy for each subject based on $K = 4$ clusters ($K$ was chosen as a cutoff point, when there was little decrease in K-means objective). We compare this approach to a case when GGMs with a static structure are estimated [5]. Note that the supervised classifiers with special EEG features are able to achieve much higher classification accuracy, however, our approach does not use any labeled data and can be seen as an exploratory step. We also used TDB-Lasso for estimating the time-varying gene networks from microarray data time series data, but due to space limit, results will be reported later in a biological paper.

Table 1: Classification accuracies based on learned brain interactions.

| Subject | aa | al | av | aw | ay |
|---------|------|------|------|------|------|
| TDB-Lasso | 0.69 | 0.80 | 0.59 | 0.67 | 0.83 |
| Static | 0.58 | 0.63 | 0.54 | 0.57 | 0.61 |

## 8 Discussion

We have developed the TDB-Lasso procedure, a novel approach for model selection and variable estimation in the varying-coefficient varying-structure models with piecewise constant functions. The VCVS models form a flexible nonparametric class of models that retain interpretability of parametric models. Due to their flexibility, important classical problems, such as linear regression with structural changes and change point detection, and some more recent problems, like structure estimation of varying graphical models, can be modeled within this class of models. The TDB-Lasso compares favorably to other commonly used [28] or latest [1] techniques for estimation in this class of models, which was demonstrated on the synthetic data. The model selection properties of the TDB-Lasso, demonstrated on the synthetic data, are also supported by the theoretical analysis. Furthermore, we demonstrate a way of applying the TDB-Lasso for graph estimation on a real dataset.

Application of the TDB-Lasso procedure goes beyond the linear varying coefficient regression models. A direct extension is to generalized varying-coefficient models $g(m(X_i, t_i)) = \mathbf{X}'_i \beta(t_i)$, $i = 1, \ldots, n$, where $g(\cdot)$ is a given link function and $m(\mathbf{X}_i, t_i) = \mathbb{E}[Y|\mathbf{X} = \mathbf{X}_i, t = t_i]$ is the conditional mean. Estimation in generalized varying-coefficient models proceeds by changing the squared loss in Eq. (7) and Eq. (8) to a different appropriate loss function. The generalized varying-coefficient models can be used to estimate the time-varying structure of discrete Markov Random Fields, again by performing the neighborhood selection.

## Footnotes

*LS is supported by a Ray and Stephenie Lane Research Fellowship. EPX is supported by grant ONR N000140910758, NSF DBI-0640543, NSF DBI-0546594, NSF IIS-0713379 and an Alfred P. Sloan Research Fellowship. We also thank Zaïd Harchaoui for useful discussions.

# References

[1] Amr Ahmed and Eric P. Xing. Tesla: Recovering time-varying networks of dependencies in social and biological studies. *Proceeding of the National Academy of Science*, 2009.

[2] Francis R. Bach. Bolasso: model consistent lasso estimation through the bootstrap. In William W. Cohen, Andrew McCallum, and Sam T. Roweis, editors, *ICML*, volume 307 of *ACM International Conference Proceeding Series*, pages 33–40. ACM, 2008.

[3] J Bai and P Perron. Computation and analysis of multiple structural change models. *Journal of Applied Econometrics*, (18):1–22, 2003.

[4] Jushan Bai and Pierre Perron. Estimating and testing linear models with multiple structural changes. *Econometrica*, 66(1):47–78, January 1998.

[5] O. Banerjee, L. El Ghaoui, and A. d'Aspremont. Model selection through sparse maximum likelihood estimation. *J. Mach. Learn. Res.*, 9:485–516, 2008.

[6] P. Bickel, Y. Ritov, and A. Tsybakov. Simultaneous analysis of lasso and dantzig selector. *Ann. of Stat.*

[7] Florentina Bunea. Honest variable selection in linear and logistic regression models via $\ell_1$ and $\ell_1 + \ell_2$ penalization. *Electronic Journal of Statistics*, 2:1153, 2008.

[8] Scott S. Chen, David L. Donoho, and Michael A. Saunders. Atomic decomposition by basis pursuit. *SIAM Journal on Scientific Computing*, 20(1):33–61, 1999.

[9] William S. Cleveland, Eric Grosse, and William M. Shyu. Local regression models. In John M. Chambers and Trevor J. Hastie, editors, *Statistical Models in S*, pages 309–376, 1991.

[10] David L. Donoho, Michael Elad, and Vladimir N. Temlyakov. Stable recovery of sparse overcomplete representations in the presence of noise. *IEEE Trans. Inform. Theory*, 52:6–18, 2006.

[11] G. Dornhege, B. Blankertz, G. Curio, and K. Müller. Boosting bit rates in non-invasive EEG single-trial classifications by feature combination and multi-class paradigms. *IEEE Trans. Biomed. Eng.*, 51:993–1002, 2004.

[12] Jianqing Fan and Qiwei Yao. *Nonlinear Time Series: Nonparametric and Parametric Methods*. (Springer Series in Statistics). Springer, August 2005.

[13] Jianqing Fan and Wenyang Zhang. Statistical estimation in varying-coefficient models. *The Annals of Statistics*, 27:1491–1518, 2000.

[14] Zaïd Harchaoui, Francis Bach, and Éric Moulines. Kernel change-point analysis. In D. Koller, D. Schuurmans, Y. Bengio, and L. Bottou, editors, *Advances in Neural Information Processing Systems 21*. 2009.

[15] Zaïd Harchaoui and Céline Levy-Leduc. Catching change-points with lasso. In J.C. Platt, D. Koller, Y. Singer, and S. Roweis, editors, *Advances in Neural Information Processing Systems 20*, pages 617–624. MIT Press, Cambridge, MA, 2008.

[16] Trevor Hastie and Robert Tibshirani. Varying-coefficient models. *Journal of the Royal Statistical Society. Series B (Methodological)*, 55(4):757–796, 1993.

[17] Mladen Kolar, Le Song, and Eric Xing. Estimating time-varying networks. In *arXiv:0812.5087*, 2008.

[18] Marc Lavielle and Eric Moulines. Least-squares estimation of an unknown number of shifts in a time series. *Journal of Time Series Analysis*, 21(1):33–59, 2000.

[19] E. Lebarbier. Detecting multiple change-points in the mean of gaussian process by model selection. *Signal Process.*, 85(4):717–736, 2005.

[20] E. Mammen and S. van de Geer. Locally adaptive regression splines. *Ann. of Stat.*, 25(1):387–413, 1997.

[21] N. Meinshausen and P. Bühlmann. High-dimensional graphs and variable selection with the lasso. *Annals of Statistics*, 34:1436, 2006.

[22] Nicolai Meinshausen and Peter Bühlmann. Stability selection. *Preprint*, 2008.

[23] Alessandro Rinaldo. Properties and refinements of the fused lasso. *Preprint*, 2008.

[24] Le Song, Mladen Kolar, and Eric P. Xing. Keller: Estimating time-evolving interactions between genes. In *Proceedings of the 16th International Conference on Intelligent Systems for Molecular Biology*, 2009.

[25] Robert Tibshirani, Michael Saunders, Saharon Rosset, Ji Zhu, and Keith Knight. Sparsity and smoothness via the fused lasso. *Journal Of The Royal Statistical Society Series B*, 67(1):91–108, 2005.

[26] S. A. van de Geer and P. Buhlmann. On the conditions used to prove oracle results for the lasso, 2009.

[27] M. J. Wainwright. Sharp thresholds for high-dimensional and noisy recovery of sparsity. *Preprint*, 2006.

[28] H. Wang and Y. Xia. Shrinkage estimation of the varying coefficient model. *Manuscript*, 2008.

[29] H Zha, C Ding, M Gu, X He, and H Simon. Spectral relaxation for k-means clustering. pages 1057–1064. MIT Press, 2001.

[30] P. Zhao and B. Yu. On model selection consistency of lasso. *J. Mach. Learn. Res.*, 7:2541–2563, 2006.

